# CONNECTIVITY VERSUS ENTROPY

Yaser S. Abu-Mostafa
*California Institute of Technology*
Pasadena, CA 91125

## ABSTRACT

How does the connectivity of a neural network (number of synapses per neuron) relate to the complexity of the problems it can handle (measured by the entropy)? Switching theory would suggest no relation at all, since all Boolean functions can be implemented using a circuit with very low connectivity (e.g., using two-input NAND gates). However, for a network that learns a problem from examples using a *local* learning rule, we prove that the entropy of the problem becomes a lower bound for the connectivity of the network.

## INTRODUCTION

The most distinguishing feature of neural networks is their ability to spontaneously learn the desired function from 'training' samples, i.e., their ability to program themselves. Clearly, a given neural network cannot just learn any function, there must be some restrictions on which networks can learn which functions. One obvious restriction, which is independent of the learning aspect, is that the network must be big enough to accommodate the circuit complexity of the function it will eventually simulate. Are there restrictions that arise merely from the fact that the network is expected to *learn* the function, rather than being purposely designed for the function? This paper reports a restriction of this kind.

The result imposes a lower bound on the connectivity of the network (number of synapses per neuron). This lower bound can only be a consequence of the learning aspect, since switching theory provides purposely designed circuits of low connectivity (e.g., using only two-input NAND gates) capable of implementing any Boolean function [1,2]. It also follows that the learning mechanism must be restricted for this lower bound to hold; a powerful mechanism can be

designed that will find one of the low-connectivity circuits (perhaps by exhaustive search), and hence the lower bound on connectivity cannot hold in general. Indeed, we restrict the learning mechanism to be local; when a training sample is loaded into the network, each neuron has access only to those bits carried by itself and the neurons it is directly connected to. This is a strong assumption that excludes sophisticated learning mechanisms used in neural-network models, but may be more plausible from a biological point of view.

The lower bound on the connectivity of the network is given in terms of the *entropy* of the environment that provides the training samples. Entropy is a quantitative measure of the disorder or randomness in an environment or, equivalently, the amount of information needed to specify the environment. There are many different ways to define entropy, and many technical variations of this concept [3]. In the next section, we shall introduce the formal definitions and results, but we start here with an informal exposition of the ideas involved.

The environment in our model produces patterns represented by $N$ bits $\mathbf{x} = x_1 \cdots x_N$ (pixels in the picture of a visual scene if you will). Only $h$ different patterns can be generated by a given environment, where $h < 2^N$ (the entropy is essentially $\log_2 h$). No knowledge is assumed about which patterns the environment is likely to generate, only that there are $h$ of them. In the learning process, a huge number of sample patterns are generated at random from the environment and input to the network, one bit per neuron. The network uses this information to set its internal parameters and gradually tune itself to this particular environment. Because of the network architecture, each neuron knows only its own bit and (at best) the bits of the neurons it is directly connected to by a synapse. Hence, the learning rules are local: a neuron does not have the benefit of the entire global pattern that is being learned.

After the learning process has taken place, each neuron is ready to perform a function *defined by what it has learned.* The collective interaction of the functions of the neurons is what defines the overall function of the network. The main result of this paper is that (roughly speaking) if the connectivity of the network is less than the entropy of the environment, the network cannot learn about the environment. The idea of the proof is to show that if the connectivity is small, the final function of each neuron is independent of the environment, and hence to conclude that the overall network has accumulated no information about the environment it is supposed to learn about.

## FORMAL RESULT

A neural network is an undirected graph (the vertices are the neurons and the edges are the synapses). Label the neurons $1, \cdots, N$ and define $K_n \subseteq \{1, \cdots, N\}$ to be the set of neurons connected by a synapse to neuron $n$, together with neuron $n$ itself. An environment is a subset $e \subseteq \{0, 1\}^N$ (each $\mathbf{x} \in e$ is a sample

from the environment). During learning, $x_1, \cdots, x_N$ (the bits of $\mathbf{x}$) are loaded into the neurons $1, \cdots, N$, respectively. Consider an arbitrary neuron $n$ and relabel everything to make $K_n$ become $\{1, \cdots, K\}$. Thus the neuron sees the first $K$ coordinates of each $\mathbf{x}$.

Since our result is asymptotic in $N$, we will specify $K$ as a function of $N$; $K = \alpha N$ where $\alpha = \alpha(N)$ satifies $\lim_{N \to \infty} \alpha(N) = \alpha_o$ ($0 < \alpha_o < 1$). Since the result is also statistical, we will consider the ensemble of environments $\mathcal{E}$

$$\mathcal{E} = \mathcal{E}(N) = \left\{ e \subseteq \{0,1\}^N \mid |e| = h \right\}$$

where $h = 2^{\beta N}$ and $\beta = \beta(N)$ satifies $\lim_{N \to \infty} \beta(N) = \beta_o$ ($0 < \beta_o < 1$). The probability distribution on $\mathcal{E}$ is uniform; any environment $e \in \mathcal{E}$ is as likely to occur as any other.

The neuron sees only the first $K$ coordinates of each $\mathbf{x}$ generated by the environment $e$. For each $e$, we define the function $n : \{0,1\}^K \to \{0,1,2,\cdots\}$ where

$$n(a_1 \cdots a_K) = |\{\mathbf{x} \in e \mid x_k = a_k \text{ for } k = 1, \cdots, K\}|$$

and the normalized version

$$\nu(a_1 \cdots a_K) = \frac{n(a_1 \cdots a_K)}{h}$$

The function $\nu$ describes the relative frequency of occurrence for each of the $2^K$ binary vectors $x_1 \cdots x_K$ as $\mathbf{x} = x_1 \cdots x_N$ runs through all $h$ vectors in $e$. In other words, $\nu$ specifies the projection of $e$ as seen by the neuron. Clearly, $\nu(\mathbf{a}) \geq 0$ for all $\mathbf{a} \in \{0,1\}^K$ and $\sum_{\mathbf{a} \in \{0,1\}^K} \nu(\mathbf{a}) = 1$.

Corresponding to two environments $e_1$ and $e_2$, we will have two functions $\nu_1$ and $\nu_2$. If $\nu_1$ is not distinguishable from $\nu_2$, the neuron cannot tell the difference between $e_1$ and $e_2$. The distinguishability between $\nu_1$ and $\nu_2$ can be measured by

$$d(\nu_1, \nu_2) = \frac{1}{2} \sum_{\mathbf{a} \in \{0,1\}^K} |\nu_1(\mathbf{a}) - \nu_2(\mathbf{a})|$$

The range of $d(\nu_1, \nu_2)$ is $0 \leq d(\nu_1, \nu_2) \leq 1$, where '0' corresponds to complete indistinguishability while '1' corresponds to maximum distinguishability. We are now in a position to state the main result.

Let $e_1$ and $e_2$ be independently selected environments from $\mathcal{E}$ according to the uniform probability distribution. $d(\nu_1, \nu_2)$ is now a random variable, and we are interested in the expected value $E(d(\nu_1, \nu_2))$. The case where $E(d(\nu_1, \nu_2)) = 0$ corresponds to the neuron getting no information about the environment, while the case where $E(d(\nu_1, \nu_2)) = 1$ corresponds to the neuron getting maximum information. The theorem predicts, in the limit, one of these extremes depending on how the connectivity ($\alpha_o$) compares to the entropy ($\beta_o$).

**Theorem.**
**1.** If $\alpha_o > \beta_o$ , then $\lim_{N \to \infty} E\left(d(\nu_1, \nu_2)\right) = 1$.
**2.** If $\alpha_o < \beta_o$ , then $\lim_{N \to \infty} E\left(d(\nu_1, \nu_2)\right) = 0$.

The proof is given in the appendix, but the idea is easy to illustrate informally. Suppose $h = 2^{K+10}$ (corresponding to part 2 of the theorem). For most environments $e \in \mathcal{E}$, the first $K$ bits of $\mathbf{x} \in e$ go through all $2^K$ possible values approximately $2^{10}$ times each as $\mathbf{x}$ goes through all $h$ possible values once. Therefore, the patterns seen by the neuron are drawn from the fixed ensemble of all binary vectors of length $K$ with essentially uniform probability distribution, i.e., $\nu$ is the same for most environments. This means that, statistically, the neuron will end up doing the same function regardless of the environment at hand.

What about the opposite case, where $h = 2^{K-10}$ (corresponding to part 1 of the theorem)? Now, with only $2^{K-10}$ patterns available from the environment, the first $K$ bits of $\mathbf{x}$ can assume at most $2^{K-10}$ values out of the possible $2^K$ values a binary vector of length $K$ can assume in principle. Furthermore, which values can be assumed depends on the particular environment at hand, i.e., $\nu$ does depend on the environment. Therefore, although the neuron still does not have the global picture, the information it has says something about the environment.

## ACKNOWLEDGEMENT

This work was supported by the Air Force Office of Scientific Research under Grant AFOSR-86-0296.

## APPENDIX

In this appendix we prove the main theorem. We start by discussing some basic properties about the ensemble of environments $\mathcal{E}$. Since the probability distribution on $\mathcal{E}$ is uniform and since $|\mathcal{E}| = \binom{2^N}{h}$, we have

$$\Pr(e) = \binom{2^N}{h}^{-1}$$

which is equivalent to generating $e$ by choosing $h$ elements $\mathbf{x} \in \{0,1\}^N$ with uniform probability (without replacement). It follows that

$$\Pr(\mathbf{x} \in e) = \frac{h}{2^N}$$

while for $x_1 \neq x_2$,

$$\Pr(x_1 \in e \ , \ x_2 \in e) = \frac{h}{2^N} \times \frac{h-1}{2^N - 1}$$

and so on.

The functions $n$ and $\nu$ are defined on $K$-bit vectors. The statistics of $n(a)$ (a random variable for fixed $a$) is independent of $a$

$$\Pr(n(a_1) = m) = \Pr(n(a_2) = m)$$

which follows from the symmetry with respect to each bit of $a$. The same holds for the statistics of $\nu(a)$. The expected value $E(n(a)) = h2^{-K}$ ($h$ objects going into $2^K$ cells), hence $E(\nu(a)) = 2^{-K}$. We now restate and prove the theorem.

**Theorem.**
**1.** If $\alpha_o > \beta_o$ , then $\lim_{N\to\infty} E\left(d(\nu_1, \nu_2)\right) = 1$.
**2.** If $\alpha_o < \beta_o$ , then $\lim_{N\to\infty} E\left(d(\nu_1, \nu_2)\right) = 0$.

**Proof.**
We expand $E\left(d(\nu_1, \nu_2)\right)$ as follows

$$
\begin{aligned}
E\left(d(\nu_1, \nu_2)\right) &= E\left(\frac{1}{2} \sum_{a \in \{0,1\}^K} |\nu_1(a) - \nu_2(a)|\right) \\
&= \frac{1}{2h} \sum_{a \in \{0,1\}^K} E\left(|n_1(a) - n_2(a)|\right) \\
&= \frac{2^K}{2h} E(|n_1 - n_2|)
\end{aligned}
$$

where $n_1$ and $n_2$ denote $n_1(0\cdots 0)$ and $n_2(0\cdots 0)$, respectively, and the last step follows from the fact that the statistics of $n_1(a)$ and $n_2(a)$ is independent of $a$. Therefore, to prove the theorem, we evaluate $E(|n_1 - n_2|)$ for large $N$.

**1.** Assume $\alpha_o > \beta_o$. Let $n$ denote $n(0\cdots 0)$, and consider $\Pr(n = 0)$. For $n$ to be zero, all $2^{N-K}$ strings $x$ of $N$ bits starting with $K$ 0's must *not* be in the environment $e$. Hence

$$\Pr(n = 0) = (1 - \frac{h}{2^N})(1 - \frac{h}{2^N - 1}) \cdots (1 - \frac{h}{2^N - 2^{N-K} + 1})$$

where the first term is the probability that $0\cdots 00 \notin e$, the second term is the

probability that $0\cdots 01 \notin e$ given that $0 \cdots 00 \notin e$, and so on.

$$\geq \left(1 - \frac{h}{2^N - 2^{N-K}}\right)^{2^{N-K}}$$

$$= \left(1 - h2^{-N}(1 - 2^{-K})^{-1}\right)^{2^{N-K}}$$

$$\geq (1 - 2h2^{-N})^{2^{N-K}}$$

$$\geq 1 - 2h2^{-N}2^{N-K}$$

$$= 1 - 2h2^{-K}$$

Hence, $\Pr(n_1 = 0) = \Pr(n_2 = 0) = \Pr(n = 0) \geq 1 - 2h2^{-K}$. However, $E(n_1) = E(n_2) = h2^{-K}$. Therefore,

$$E(|n_1 - n_2|) = \sum_{i=0}^{h}\sum_{j=0}^{h} \Pr(n_1 = i, n_2 = j)|i - j|$$

$$= \sum_{i=0}^{h}\sum_{j=0}^{h} \Pr(n_1 = i)\Pr(n_2 = j)|i - j|$$

$$\geq \sum_{j=0}^{h} \Pr(n_1 = 0)\Pr(n_2 = j)j$$

$$+ \sum_{i=0}^{h} \Pr(n_1 = i)\Pr(n_2 = 0)i$$

which follows by throwing away all the terms where neither $i$ nor $j$ is zero (the term where both $i$ an $j$ are zero appears twice for convenience, but this term is zero anyway).

$$= \Pr(n_1 = 0)E(n_2) + \Pr(n_2 = 0)E(n_1)$$

$$\geq 2(1 - 2h2^{-K})h2^{-K}$$

Substituting this estimate in the expression for $E(d(\nu_1, \nu_2))$, we get

$$E(d(\nu_1, \nu_2)) = \frac{2^K}{2h}E(|n_1 - n_2|)$$

$$\geq \frac{2^K}{2h} \times 2(1 - 2h2^{-K})h2^{-K}$$

$$= 1 - 2h2^{-K}$$

$$= 1 - 2 \times 2^{(\beta - \alpha)N}$$

Since $\alpha_o > \beta_o$ by assumption, this lower bound goes to 1 as $N$ goes to infinity. Since 1 is also an upper bound for $d(\nu_1, \nu_2)$ (and hence an upper bound for the expected value $E(d(\nu_1, \nu_2))$), $\lim_{N\to\infty} E(d(\nu_1, \nu_2))$ must be 1.

**2.** Assume $\alpha_o < \beta_o$. Consider

$$
\begin{aligned}
E(|n_1 - n_2|) &= E\left(|(n_1 - h2^{-K}) - (n_2 - h2^{-K})|\right) \\
&\leq E(|n_1 - h2^{-K}| + |n_2 - h2^{-K}|) \\
&= E(|n_1 - h2^{-K}|) + E(|n_2 - h2^{-K}|) \\
&= 2E(|n - h2^{-K}|)
\end{aligned}
$$

To evaluate $E(|n - h2^{-K}|)$, we estimate the variance of $n$ and use the fact that $E(|n - h2^{-K}|) \leq \sqrt{\text{var}(n)}$ (recall that $h2^{-K} = E(n)$). Since $\text{var}(n) = E(n^2) - (E(n))^2$, we need an estimate for $E(n^2)$. We write $n = \sum_{\mathbf{a} \in \{0,1\}^{N-K}} \delta_{\mathbf{a}}$, where

$$
\delta_{\mathbf{a}} = \begin{cases} 1, & \text{if } 0\cdots 0\mathbf{a} \in e; \\ 0, & \text{otherwise.} \end{cases}
$$

In this notation, $E(n^2)$ can be written as

$$
\begin{aligned}
E(n^2) &= E\left( \sum_{\mathbf{a} \in \{0,1\}^{N-K}} \sum_{\mathbf{b} \in \{0,1\}^{N-K}} \delta_{\mathbf{a}} \delta_{\mathbf{b}} \right) \\
&= \sum_{\mathbf{a} \in \{0,1\}^{N-K}} \sum_{\mathbf{b} \in \{0,1\}^{N-K}} E(\delta_{\mathbf{a}} \delta_{\mathbf{b}})
\end{aligned}
$$

For the 'diagonal' terms $(\mathbf{a} = \mathbf{b})$,

$$
\begin{aligned}
E(\delta_{\mathbf{a}} \delta_{\mathbf{a}}) &= \Pr(\delta_{\mathbf{a}} = 1) \\
&= h2^{-N}
\end{aligned}
$$

There are $2^{N-K}$ such diagonal terms, hence a total contribution of $2^{N-K} \times h2^{-N} = h2^{-K}$ to the sum. For the 'off-diagonal' terms $(\mathbf{a} \neq \mathbf{b})$,

$$
\begin{aligned}
E(\delta_{\mathbf{a}} \delta_{\mathbf{b}}) &= \Pr(\delta_{\mathbf{a}} = 1, \delta_{\mathbf{b}} = 1) \\
&= \Pr(\delta_{\mathbf{a}} = 1)\Pr(\delta_{\mathbf{b}} = 1|\delta_{\mathbf{a}} = 1) \\
&= \frac{h}{2^N} \times \frac{h-1}{2^N - 1}
\end{aligned}
$$

There are $2^{N-K}(2^{N-K}-1)$ such off-diagonal terms, hence a total contribution of $2^{N-K}(2^{N-K}-1) \times \frac{h(h-1)}{2^N(2^N-1)} \leq (h2^{-K})^2 \frac{2^N}{2^N-1}$ to the sum. Putting the contributions

from the diagonal and off-diagonal terms together, we get

$$E(n^2) \leq h2^{-K} + (h2^{-K})^2 \frac{2^N}{2^N - 1}$$

$$\text{var}(n) = E(n^2) - (E(n))^2$$

$$\leq \left( h2^{-K} + (h2^{-K})^2 \frac{2^N}{2^N - 1} \right) - \left( h2^{-K} \right)^2$$

$$= h2^{-K} + (h2^{-K})^2 \frac{1}{2^N - 1}$$

$$= h2^{-K} \left( 1 + \frac{h2^{-K}}{2^N - 1} \right)$$

$$\leq 2h2^{-K}$$

The last step follows since $h2^{-K}$ is much smaller than $2^N - 1$. Therefore, $E(|n - h2^{-K}|) \leq \sqrt{\text{var}(n)} \leq \left( 2h2^{-K} \right)^{\frac{1}{2}}$. Substituting this estimate in the expression for $E(d(\nu_1, \nu_2))$, we get

$$E(d(\nu_1, \nu_2)) = \frac{2^K}{2h} E(|n_1 - n_2|)$$

$$\leq \frac{2^K}{2h} \times 2E(|n - h2^{-K}|)$$

$$\leq \frac{2^K}{2h} \times 2 \times \left( 2h2^{-K} \right)^{\frac{1}{2}}$$

$$= \left( 2\frac{2^K}{h} \right)^{\frac{1}{2}}$$

$$= \sqrt{2} \times 2^{\frac{1}{2}(\alpha - \beta)N}$$

Since $\alpha_o < \beta_o$ by assumption, this upper bound goes to 0 as $N$ goes to infinity. Since 0 is also a lower bound for $d(\nu_1, \nu_2)$ (and hence a lower bound for the expected value $E(d(\nu_1, \nu_2))$), $\lim_{N \to \infty} E(d(\nu_1, \nu_2))$ must be 0. ∎

## REFERENCES

[1] Y. Abu-Mostafa, "Neural networks for computing?," *AIP Conference Proceedings # 151, Neural Networks for Computing*, J. Denker (ed.), pp. 1-6, 1986.

[2] Z. Kohavi, *Switching and Finite Automata Theory*, McGraw-Hill, 1978.

[3] Y. Abu-Mostafa, "The complexity of information extraction," *IEEE Trans. on Information Theory*, vol. IT-32, pp. 513-525, July 1986.

[4] Y. Abu-Mostafa, "Complexity in neural systems," in *Analog VLSI and Neural Systems* by C. Mead, Addison-Wesley, 1988.
